# Single Channel Speech Separation Using Factorial Dynamics

**John R. Hershey**          **Trausti Kristjansson**          **Steven Rennie**          **Peder A. Olsen**

**IBM Thomas J. Watson Research Center**
Yorktown Heights, NY 10598

## Abstract

Human listeners have the extraordinary ability to hear and recognize speech even when more than one person is talking. Their machine counterparts have historically been unable to compete with this ability, until now. We present a model-based system that performs on par with humans in the task of separating speech of two talkers from a single-channel recording. Remarkably, the system surpasses human recognition performance in many conditions. The models of speech use temporal dynamics to help infer the source speech signals, given mixed speech signals. The estimated source signals are then recognized using a conventional speech recognition system. We demonstrate that the system achieves its best performance when the model of temporal dynamics closely captures the grammatical constraints of the task.

One of the hallmarks of human perception is our ability to solve the auditory cocktail party problem: we can direct our attention to a given speaker in the presence of interfering speech, and understand what was said remarkably well. Until now the same could not be said for automatic speech recognition systems. However, we have recently introduced a system which in many conditions performs this task better than humans [1][2]. The model addresses the Pascal Speech Separation Challenge task [3], and outperforms all other published results by more than 10% word error rate (WER). In this model, dynamics are modeled using a layered combination of one or two Markov chains: one for long-term dependencies and another for short-term dependencies. The combination of the two speakers was handled via an iterative Laplace approximation method known as Algonquin [4]. Here we describe experiments that show better performance on the same task with a simpler version of the model.

The task we address is provided by the PASCAL Speech Separation Challenge [3], which provides standard training, development, and test data sets of single-channel speech mixtures following an arbitrary but simple grammar. In addition, the challenge organizers have conducted human-listening experiments to provide an interesting baseline for comparison of computational techniques.

The overall system we developed is composed of the three components: a speaker identification and gain estimation component, a signal separation component, and a speech recognition system. In this paper we focus on the signal separation component, which is composed of the acoustic and grammatical models. The details of the other components are discussed in [2].

Single-channel speech separation has previously been attempted using Gaussian mixture models (GMMs) on individual frames of acoustic features. However such models tend to perform well only when speakers are of different gender or have rather different voices [4]. When speakers have similar voices, speaker-dependent mixture models cannot unambiguously identify the component speakers. In such cases it is helpful to model the temporal dynamics of the speech. Several models in the literature have attempted to do so either for recognition [5, 6] or enhancement [7, 8] of speech. Such

models have typically been based on a discrete-state hidden Markov model (HMM) operating on a frame-based acoustic feature vector.

Modeling the dynamics of the log spectrum of speech is challenging in that different speech components evolve at different time-scales. For example the excitation, which carries mainly pitch, versus the filter, which consists of the formant structure, are somewhat independent of each other. The formant structure closely follows the sequences of phonemes in each word, which are pronounced at a rate of several per second. In non-tonal languages such as English, the pitch fluctuates with prosody over the course of a sentence, and is not directly coupled with the words being spoken. Nevertheless, it seems to be important in separating speech, because the pitch harmonics carry predictable structure that stands out against the background.

We address the various dynamic components of speech by testing different levels of dynamic constraints in our models. We explore four different levels of dynamics: *no dynamics*, low-level *acoustic dynamics*, high-level *grammar dynamics*, and a layered combination, *dual dynamics*, of the acoustic and grammar dynamics. The grammar dynamics and dual dynamics models perform the best in our experiments.

The acoustic models are combined to model mixtures of speech using two methods: a nonlinear model known as *Algonquin*, which models the combination of log-spectrum models as a sum in the power spectrum, and a simpler *max* model that combines two log spectra using the max function. It turns out that whereas Algonquin works well, our formulation of the max model does better overall.

With the combination of the max model and grammar-level dynamics, the model produces remarkable results: it is often able to extract two utterances from a mixture even when they are from the same speaker [1]. Overall results are given in Table 1, which shows that our closest competitors are human listeners.

Table 1: Overall word error rates across all conditions on the challenge task. *Human*: average human error rate, *IBM*: our best result, *Next Best*: the best of the eight other published results on this task, and *Chance*: the theoretical error rate for random guessing.

| **System**: | Human | IBM | Next Best | Chance |
|---|---|---|---|---|
| **Word Error Rate**: | 22.3% | 22.6% | 34.2% | 93.0% |

# 1 Speech Models

The model consists of an *acoustic model* and *temporal dynamics model* for each source, and a *mixing model*, which models how the source models are combined to describe the mixture. The acoustic features were short-time log spectrum frames computed every 15 ms. Each frame was of length 40 ms and a 640-point mixed-radix FFT was used. The DC component was discarded, producing a 319-dimensional log-power-spectrum feature vector $\mathbf{y}_t$.

The acoustic model consists of a set of diagonal-covariance Gaussians in the features. For a given speaker, $a$, we model the conditional probability of the log-power spectrum of each source signal $\mathbf{x}^a$ given a discrete acoustic state $s^a$ as Gaussian, $p(\mathbf{x}^a|s^a) = N(\mathbf{x}^a; \mu_{s^a}, \mathbf{\Sigma}_{s^a})$, with mean $\mu_{s^a}$, and covariance matrix $\mathbf{\Sigma}_{s^a}$. We used 256 Gaussians, one per acoustic state, to model the acoustic space of each speaker. For efficiency and tractability we restrict the covariance to be diagonal. A model with no dynamics can be formulated by producing state probabilities $p(s^a)$, and is depicted in 1(a).

**Acoustic Dynamics**: To capture the low-level dynamics of the acoustic signal, we modeled the acoustic dynamics of a given speaker, $a$, via state transitions $p(s_t^a|s_{t-1}^a)$ as shown in Figure 1(b). There are 256 acoustic states, hence for each speaker $a$, we estimated a $256 \times 256$ element transition matrix $A^a$.

**Grammar Dynamics**: The grammar dynamics are modeled by grammar state transitions, $p(v_t^a|v_{t-1}^a)$, which consist of left-to-right phone models. The legal word sequences are given by the Speech Separation Challenge grammar [3] and are modeled using a set of pronunciations that

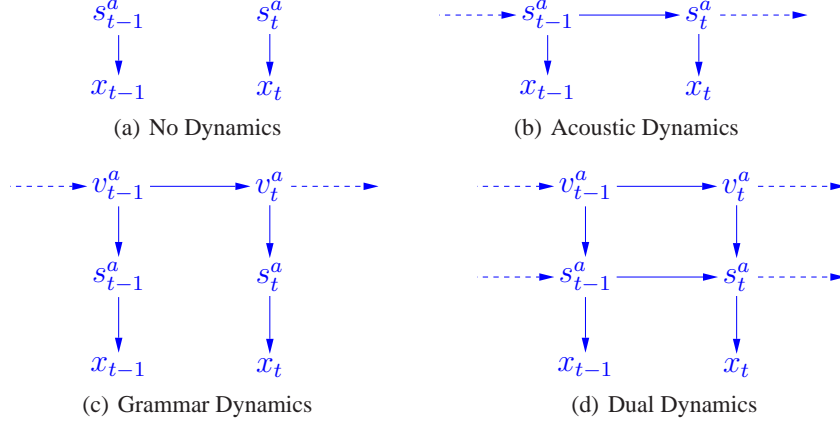

(a) No Dynamics    (b) Acoustic Dynamics

(c) Grammar Dynamics    (d) Dual Dynamics

Figure 1: Graph of models for a given source. In (a), there are no dynamics, so the model is a simple mixture model. In (b), only acoustic dynamics are modeled. In (c), grammar dynamics are modeled with a shared set of acoustic Gaussians, in (d) dual – grammar and acoustic – dynamics have been combined. Note that (a) (b) and (c) are special cases of (d), where different nodes are assumed independent.

map from words to three-state context-dependent phone models. The state transition probabilities derived from these phone models are sparse in the sense that most transition probabilities are zero.

We model speaker dependent distributions $p(s^a|v^a)$ that associate the grammar states, $v^a$ to the speaker-dependent acoustic states. These are learned from training data where the grammar state sequences and acoustic state sequences are known for each utterance. The grammar of our system has 506 states, so we estimate a $506 \times 256$ element conditional probability matrix $B^a$ for each speaker.

**Dual Dynamics**: The dual-dynamics model combines the acoustic dynamics with the grammar dynamics. It is useful in this case to avoid modeling the full combination of $s$ and $v$ states in the joint transitions $p(s_t^a|s_{t-1}^a, v_t)$. Instead we make a naive-Bayes assumption to approximate this as $\frac{1}{z}p(s_t^a|s_{t-1}^a)^\alpha p(s_t^a|v_t)^\beta$, where $\alpha$ and $\beta$ adjust the relative influence of the two probabilities, and $z$ is the normalizing constant. Here we simply use the probability matrices $A^a$ and $B^a$, defined above.

## 2  Mixed Speech Models

The speech separation challenge involves recognizing speech in mixtures of signals from two speakers, $a$ and $b$. We consider only mixing models that operate independently on each frequency for analytical and computational tractability. The short-time log spectrum of the mixture $y_t$, in a given frequency band, is related to that of the two sources $x_t^a$ and $x_t^b$ via the *mixing model* given by the conditional probability distribution, $p(y|x^a, x^b)$. The joint distribution of the observation and source in one feature dimension, given the source states is thus:

$$p(y_t, x_t^a, x_t^b|s_t^a, s_t^b) = p(y_t|x_t^a, x_t^b)p(x_t^a|s_t^a)p(x_t^b|s_t^b). \tag{1}$$

In general, to infer and reconstruct speech we need to compute the likelihood of the observed mixture

$$p(y_t|s_t^a, s_t^b) = \int p(y_t, x_t^a, x_t^b|s_t^a, s_t^b)dx_t^a dx_t^b, \tag{2}$$

and the posterior expected values of the sources given the states,

$$E(x_t^a|y_t, s_t^a, s_t^b) = \int x_t^a p(x_t^a, x_t^b|y_t, s_t^a, s_t^b)dx_t^a dx_t^b, \tag{3}$$

and similarly for $x_t^b$. These quantities, combined with a prior model for the joint state sequences $\{s_{1..T}^a, s_{1..T}^b\}$, allow us to compute the minimum mean squared error (MMSE) estimators $E(\mathbf{x}_{1..T}^a|\mathbf{y}_{1..T})$ or the maximum *a posteriori* (MAP) estimate $E(\mathbf{x}_{1..T}^a|y_{1..T}, \hat{s}^a_{1..T}, \hat{s}^b_{1..T})$,

where $\hat{s}^a{}_{1..T}, \hat{s}^b{}_{1..T} = \arg\max_{s^a_{1..T}, s^b_{1..T}} p(s^a_{1..T}, s^b_{1..T}|\mathbf{y}_{1..T})$, where the subscript, $1..T$, refers to all frames in the signal.

The mixing model can be defined in a number of ways. We explore two popular candidates, for which the above integrals can be readily computed: *Algonquin*, and the *max model*.

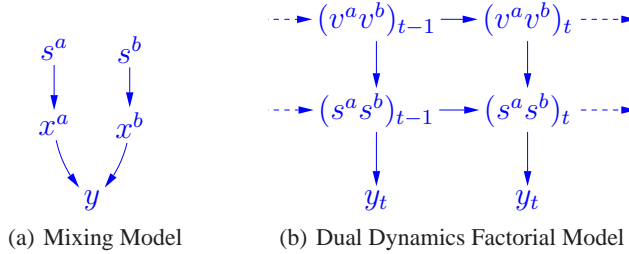

(a) Mixing Model    (b) Dual Dynamics Factorial Model

Figure 2: Model combination for two talkers. In (a) all dependencies are shown. In (b) the full dual-dynamics model is graphed with the $\mathbf{x}^a$ and $\mathbf{x}^b$ integrated out, and corresponding states from each speaker combined into product states. The other models are special cases of this graph with different edges removed, as in Figure 1.

**Algonquin**: The relationship between the sources and mixture in the log power spectral domain is approximated as
$$p(y_t|x^a_t, x^b_t) = N(y_t; \log(\exp(x^a_t) + \exp(x^b_t)), \Psi) \qquad (4)$$
where $\Psi$ is introduced to model the error due to the omission of phase [4]. An iterative Newton-Laplace method accurately approximates the conditional posterior $p(x^a_t, x^b_t|y_t, s^a_t, s^b_t)$ from (1) as Gaussian. This Gaussian allows us to analytically compute the observation likelihood $p(y_t|s^a_t, s^b_t)$ and expected value $E(x^a_t|y_t, s^a_t, s^b_t)$, as in [4].

**Max model**: The mixing model is simplified using the fact that log of a sum is approximately the log of the maximum:
$$p(y|x^a, x^b) = \delta\left(y - \max(x^a, x^b)\right) \qquad (5)$$
In this model the likelihood is
$$p(y_t|s^a_t, s^b_t) = p_{x^a_t}(y_t|s^a_t)\Phi_{x^b}(y_t|s^b_t) + p_{x^b_t}(y_t|s^b_t)\Phi_{x^a}(y_t|s^a_t), \qquad (6)$$
where $\Phi_{x^a_t}(y_t|s^a_t) = \int_{-\infty}^{y_t} N(x^a_t; \mu_{s^a_t}, \Sigma_{s^a_t})dx^a_t$ is a Gaussian cumulative distribution function [5]. In [5], such a model was used to compute state likelihoods and find the optimal state sequence. In [8], a simplified model was used to infer binary masking values for refiltering.

We take the max model a step further and derive source posteriors, so that we can compute the MMSE estimators for the log power spectrum. Note that the source posteriors in $x^a_t$ and $x^b_t$ are each a mixture of a delta function and a truncated Gaussian. Thus we analytically derive the necessary expected value:
$$E(x^a_t|y_t, s^a_t, s^b_t) = p(x^a_t = y_t|y_t, s^a_t, s^b_t)y_t + p(x^a_t < y_t|y_t, s^a_t, s^b_t)E(x^a_t|x^a_t < y_t, s^a_t) \qquad (7)$$
$$= \pi^a_t y_t + \pi^b_t \left(\mu_{s^a_t} - \Sigma_{s^a_t}\frac{p_{x^a_t}(y_t|s^a_t)}{\Phi_{x^a_t}(y_t|s^a_t)}\right), \qquad (8)$$
with weights $\pi^a_t = p(x^a_t = y_t|y_t, s^a_t, s^b_t) = p_{x^a_t}(y_t|s^a_t)\Phi_{x^b}(y_t|s^b_t)/p(y_t|s^a_t, s^b_t)$, and $\pi^b_t = 1 - \pi^a_t$. For many pairs of states one model is significantly louder than another $\mu_{s^a} \gg \mu_{s^b}$ in a given frequency band, relative to their variances. In such cases it is reasonable to approximate the likelihood as $p(y_t|s^a_t, s^b_t) \approx p_{x^a_t}(y_t|s^a_t)$, and the posterior expected values according to $E(x^a_t|y_t, s^a_t, s^b_t) \approx y_t$ and $E(x^b_t|y_t, s^a_t, s^b_t) \approx \min(y_t, \mu_{s^b_t})$, and similarly for $\mu_{s^a} \ll \mu_{s^b}$.

## 3   Likelihood Estimation

Because of the large number of state combinations, the model would not be practical without techniques to reduce computation time. To speed up the evaluation of the joint state likelihood, we employed both *band quantization* of the acoustic Gaussians and *joint-state pruning*.

**Band Quantization**: One source of computational savings stems from the fact that some of the Gaussians in our model may differ only in a few features. Band quantization addresses this by approximating each of the $D$ Gaussians of each model with a shared set of $d$ Gaussians, where $d \ll D$, in each of the $F$ frequency bands of the feature vector. A similar idea is described in [9]. It relies on the use of a diagonal covariance matrix, so that $p(\mathbf{x}^a|s^a) = \prod_f N(x_f^a; \mu_{f,s^a}, \Sigma_{f,s^a})$, where $\Sigma_{f,s^a}$ are the diagonal elements of covariance matrix $\mathbf{\Sigma}_{s^a}$. The mapping $M_f(s^i)$ associates each of the $D$ Gaussians with one of the $d$ Gaussians in band $f$. Now $\hat{p}(x^a|s^a) = \prod_f N(x_f^a; \mu_{f,M_f(s^a)}, \Sigma_{f,M_f(s^a)})$ is used as a surrogate for $p(x^a|s^a)$. Figure 3 illustrates the idea.

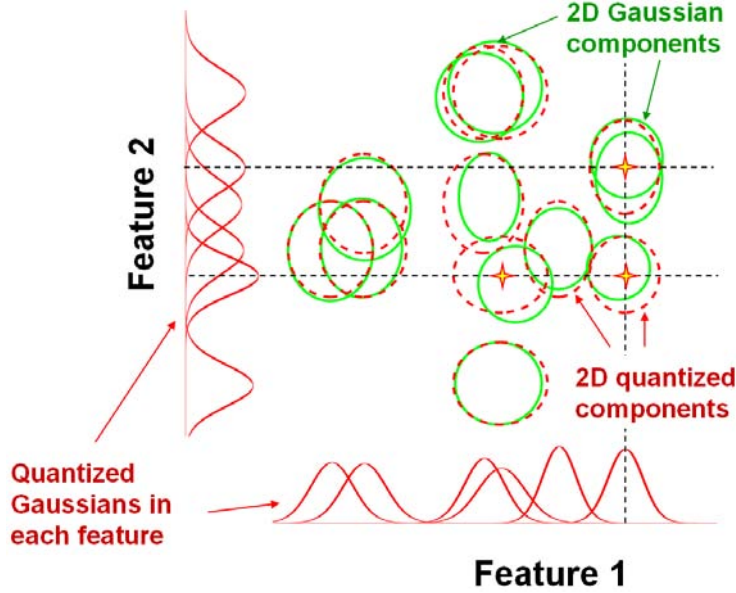

Figure 3: In band quantization, many multi-dimensional Gaussians are mapped to a few unidimensional Gaussians.

Under this model the $d$ Gaussians are optimized by minimizing the KL-divergence $D(\sum_{s^a} p(s^a)p(x^a|s^a) || \sum_{s^a} p(s^a)\hat{p}(x^a|s^a))$, and likewise for $s^b$. Then in each frequency band, only $d \times d$, instead of $D \times D$ combinations of Gaussians have to be evaluated to compute $p(y|s^a, s^b)$. Despite the relatively small number of components $d$ in each band, taken across bands, band quantization is capable of expressing $d^F$ distinct patterns, in an $F$-dimensional feature space, although in practice only a subset of these will be used to approximate the Gaussians in a given model. We used $d = 8$ and $D = 256$, which reduced the likelihood computation time by three orders of magnitude.

**Joint State Pruning**: Another source of computational savings comes from the sparseness of the model. Only a handful of $s^a$, $s^b$ combinations have likelihoods that are significantly larger than the rest for a given observation. Only these states are required to adequately explain the observation. By pruning the total number of combinations down to a smaller number we can speed up the likelihood calculation, estimation of the components signals, as well as the temporal inference.

However, we must estimate the likelihoods in order to determine which states to retain. We therefore used band-quantization to estimate likelihoods for all states, perform state pruning, and then the full model on the pruned states using the exact parameters. In the experiments reported here, we pruned down to 256 state combinations. The effect of these speedup methods on accuracy will be reported in a future publication.

## 4 Inference

In our experiments we performed inference in four different conditions: *no dynamics*, with *acoustic dynamics* only, with *grammar dynamics* only, and with *dual dynamics* (acoustic and grammar). With no dynamics the source models reduce to GMMs and we infer MMSE estimates of the sources based

on $p(x^a, x^b|y)$ as computed from (1), using Algonquin or the max model. Once the log spectrum of each source is estimated, we estimate the corresponding time-domain signal as shown in [4].

In the acoustic dynamics condition the exact inference algorithm uses a 2-Dimensional Viterbi search, described below, with acoustic temporal constraints $p(s_t|s_{t-1})$ and likelihoods from Eqn. (1), to find the most likely joint state sequence $s_{1..T}$. Similarly in the grammar dynamics condition, 2-D Viterbi search is used to infer the grammar state sequences, $v_{1..T}$. Instead of single Gaussians as the likelihood models, however, we have mixture models in this case. So we can perform an MMSE estimate of the sources by averaging over the posterior probability of the mixture components given the grammar Viterbi sequence, and the observations.

It is critical to use the 2-D Viterbi algorithm in both cases, rather than the forward-backward algorithm, because in the same-speaker condition at 0dB, the acoustic models and dynamics are symmetric. This symmetry means that the posterior is essentially bimodal and averaging over these modes would yield identical estimates for both speakers. By finding the best path through the joint state space, the 2-D Viterbi algorithm breaks this symmetry and allows the model to make different estimates for each speaker.

In the dual-dynamics condition we use the model of section 2(b). With two speakers, exact inference is computationally complex because the full joint distribution of the grammar and acoustic states, $(v^a \times s^a) \times (v^b \times s^b)$ is required and is very large in number. Instead we perform approximate inference by alternating the 2-D Viterbi search between two factors: the Cartesian product $s^a \times s^b$ of the acoustic state sequences and the Cartesian product $v^a \times v^b$ of the grammar state sequences. When evaluating each state sequence we hold the other chain constant, which decouples its dynamics and allows for efficient inference. This is a useful factorization because the states $s^a$ and $s^b$ interact strongly with each other and similarly for $v^a$ and $v^b$. Again, in the same-talker condition, the 2-D Viterbi search breaks the symmetry in each factor.

**2-D Viterbi search**: The Viterbi algorithm estimates the maximum-likelihood state sequence $s_{1..T}$ given the observations $x_{1..T}$. The complexity of the Viterbi search is $O(TD^2)$ where $D$ is the number of states and $T$ is the number of frames. For producing MAP estimates of the 2 sources, we require a 2 dimensional Viterbi search which finds the most likely joint state sequences $s_{1..T}^a$ and $s_{1..T}^b$ given the mixed signal $y_{1..T}$ as was proposed in [5].

On the surface, the 2-D Viterbi search appears to be of complexity $O(TD^4)$. Surprisingly, it can be computed in $O(TD^3)$ operations. This stems from the fact that the dynamics for each chain are independent. The forward-backward algorithm for a factorial HMM with $N$ state variables requires only $O(TND^{N+1})$ rather than the $O(TD^{2N})$ required for a naive implementation [10]. The same is true for the Viterbi algorithm. In the Viterbi algorithm, we wish to find the most probable paths leading to each state by finding the two arguments $s_{t-1}^a$ and $s_{t-1}^b$ of the following maximization:

$$\{\hat{s}_{t-1}^a, \hat{s}_{t-1}^b\} = \arg\max_{s_{t-1}^a s_{t-1}^b} p(s_t^a|s_{t-1}^a)p(s_t^b|s_{t-1}^b)p(s_{t-1}^a, s_{t-1}^b|y_{1..t-1})$$

$$= \arg\max_{s_{t-1}^a} p(s_t^a|s_{t-1}^a) \max_{s_{t-1}^b} p(s_t^b|s_{t-1}^b)p(s_{t-1}^a, s_{t-1}^b|y_{1..t-1}). \qquad (9)$$

The two maximizations can be done in sequence, requiring $O(D^3)$ operations with $O(D^2)$ storage for each step. In general, as with the forward-backward algorithm, the $N$-dimensional Viterbi search requires $O(TND^{N+1})$ operations.

We can also exploit the sparsity of the transition matrices and observation likelihoods, by pruning unlikely values. Using both of these methods our implementation of 2-D Viterbi search is faster than the acoustic likelihood computation that serves as its input, for the model sizes and grammars chosen in the speech separation task.

**Speaker and Gain Estimation**: In the challenge task, the gains and identities of the two speakers were unknown at test time and were selected from a set of $34$ speakers which were mixed at SNRs ranging from 6dB to -9dB. We used speaker-dependent acoustic models because of their advantages when separating different speakers. These models were trained on gain-normalized data, so the models are not well matched to the different gains of the signals at test time. This means that we have to estimate both the speaker identities and the gain in order to adapt our models to the source signals for each test utterance.

The number of speakers and range of SNRs in the test set makes it too expensive to consider every possible combination of models and gains. Instead, we developed an efficient model-based method for identifying the speakers and gains, described in [2]. The algorithm is based upon a very simple idea: identify and utilize frames that are dominated by a single source – based on their likelihoods under each speaker-dependent acoustic model – to determine what sources are present in the mixture. Using this criteria we can eliminate most of the unlikely speakers, and explore all combinations of the remaining speakers. An approximate EM procedure is then used to select a single pair of speakers and estimate their gains.

**Recognition**: Although inference in the system may involve recognition of the words– for models that contain a grammar –we still found that a separately trained recognizer performed better. After reconstruction, each of the two signals is therefore decoded with a speech recognition system that incorporates Speaker Dependent Labeling (SDL) [2].

This method uses speaker dependent models for each of the 34 speakers. Instead of using the speaker identities provided by the speaker ID and gain module, we followed the approach for gender dependent labeling (GDL) described in [11]. This technique provides better results than if the true speaker ID is specified.

## 5   Results

The Speech Separation Challenge [3] involves separating the mixed speech of two speakers drawn from of a set of 34 speakers. An example utterance is *place white by R 4 now*. In each recording, one of the speakers says *white* while the other says *blue*, *red* or *green*. The task is to recognize the letter and the digit of the speaker that said *white*. Using the SDL recognizer, we decoded the two estimated signals under the assumption that one signal contains white and the other does not, and vice versa. We then used the association that yielded the highest combined likelihood.

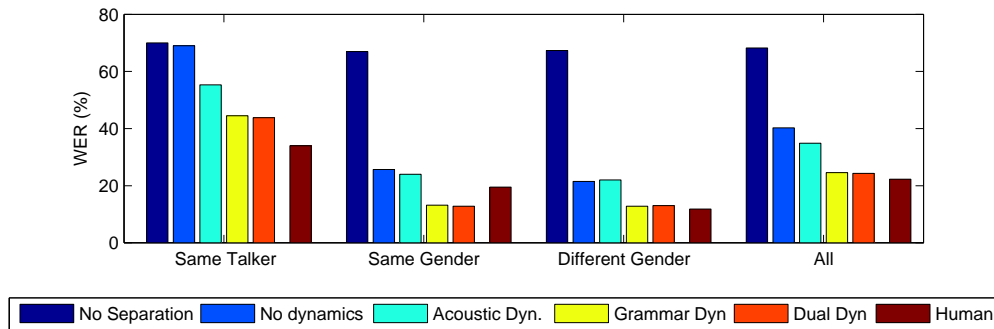

Figure 4: Average word error rate (WER) as a function of model dynamics, in different talker conditions, compared to Human error rates, using Algonquin.

Human listener performance [3] is compared in Figure 4 to results using the SDL recognizer without speech separation, and for each the proposed models. Performance is poor without separation in all conditions. With no dynamics the models do surprisingly well in the different talker conditions, but poorly when the signals come from the same talker. Acoustic dynamics gives some improvement, mainly in the same-talker condition. The grammar dynamics seems to give the most benefit, bringing the error rate in the same-gender condition below that of humans. The dual-dynamics model performed about the same as the grammar dynamics model, despite our intuitions. Replacing Algonquin with the max model reduced the error rate in the dual dynamics model (from 24.3% to 23.5%) and grammar dynamics model (from 24.6% to 22.6%), which brings the latter closer than any other model to the human recognition rate of 22.3%.

Figure 5 shows the relative word error rate of the best system compared to human subjects. When both speakers are around the same loudness, the system exceeds human performance, and in the same-gender condition makes less than half the errors of the humans. Human listeners do better when the two signals are at different levels, even if the target is below the masker (i.e., in -9dB), suggesting that they are better able to make use of differences in amplitude as a cue for separation.

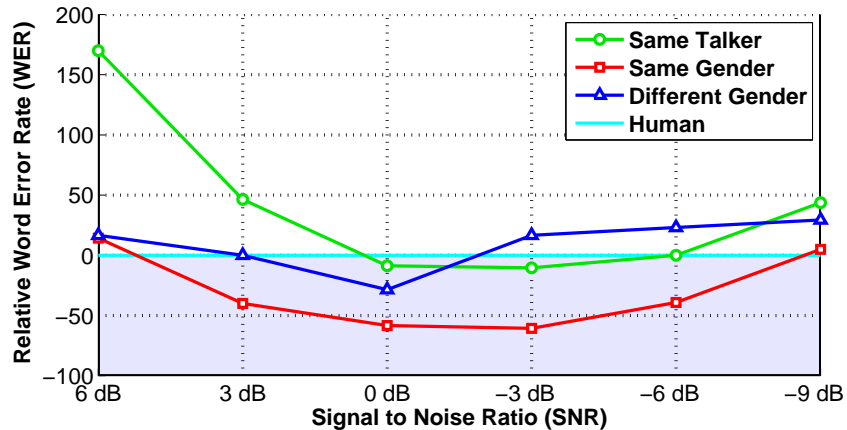

Figure 5: Word error rate of best system relative to human performance. Shaded area is where the system outperforms human listeners.

An interesting question is to what extent different grammar constraints affect the results. To test this, we limited the grammar to just the two test utterances, and the error rate on the estimated sources dropped to around 10%. This may be a useful paradigm for separating speech from background noise when the text is known, such as in closed-captioned recordings. At the other extreme, in realistic speech recognition scenarios, there is little knowledge of the background speaker's grammar. In such cases the benefits of models of low-level acoustic continuity over purely grammar-based systems may be more apparent.

It is our hope that further experiments with both human and machine listeners will provide us with a better understanding of the differences in their performance characteristics, and provide insights into how the human auditory system functions, as well as how automatic speech perception in general can be brought to human levels of performance.

## Footnotes

[1]Demos and information can be found at: `http : //www.research.ibm.com/speechseparation`

## References

[1] T. Kristjansson, J. R. Hershey, P. A. Olsen, S. Rennie, and R. Gopinath, "Super-human multi-talker speech recognition: The IBM 2006 speech separation challenge system," in *ICSLP*, 2006.

[2] Steven Rennie, Pedera A. Olsen, John R. Hershey, and Trausti Kristjansson, "Separating multiple speakers using temporal constraints," in *ISCA Workshop on Statistical And Perceptual Audition*, 2006.

[3] Martin Cooke and Tee-Won Lee, "Interspeech speech separation challenge," http://www.dcs.shef.ac.uk/∼martin/SpeechSeparationChallenge.htm, 2006.

[4] T. Kristjansson, J. Hershey, and H. Attias, "Single microphone source separation using high resolution signal reconstruction," *ICASSP*, 2004.

[5] P. Varga and R.K. Moore, "Hidden Markov model decomposition of speech and noise," *ICASSP*, pp. 845–848, 1990.

[6] M. Gales and S. Young, "Robust continuous speech recognition using parallel model combination," *IEEE Transactions on Speech and Audio Processing*, vol. 4, no. 5, pp. 352–359, September 1996.

[7] Y. Ephraim, "A Bayesian estimation approach for speech enhancement using hidden Markov models.," vol. 40, no. 4, pp. 725–735, 1992.

[8] S. Roweis, "Factorial models and refiltering for speech separation and denoising," *Eurospeech*, pp. 1009–1012, 2003.

[9] E. Bocchieri, "Vector quantization for the efficient computation of continuous density likelihoods. proceedings of the international conference on acoustics," in *ICASSP*, 1993, vol. II, pp. 692–695.

[10] Zoubin Ghahramani and Michael I. Jordan, "Factorial hidden Markov models," in *Advances in Neural Information Processing Systems*, vol. 8.

[11] Peder Olsen and Satya Dharanipragada, "An efficient integrated gender detection scheme and time mediated averaging of gender dependent acoustic models," in *Eurospeech 2003*, 2003, vol. 4, pp. 2509–2512.
